# Boltzmann Machine learning using mean field theory and linear response correction

**H.J. Kappen**
Department of Biophysics
University of Nijmegen, Geert Grooteplein 21
NL 6525 EZ Nijmegen, The Netherlands

**F. B. Rodríguez**
Instituto de Ingeniería del Conocimiento & Departamento de Ingeniería Informática,
Universidad Autónoma de Madrid, Canto Blanco,28049 Madrid, Spain

## Abstract

We present a new approximate learning algorithm for Boltzmann Machines, using a systematic expansion of the Gibbs free energy to second order in the weights. The linear response correction to the correlations is given by the Hessian of the Gibbs free energy. The computational complexity of the algorithm is cubic in the number of neurons. We compare the performance of the exact BM learning algorithm with first order (Weiss) mean field theory and second order (TAP) mean field theory. The learning task consists of a fully connected Ising spin glass model on 10 neurons. We conclude that 1) the method works well for paramagnetic problems 2) the TAP correction gives a significant improvement over the Weiss mean field theory, both for paramagnetic and spin glass problems and 3) that the inclusion of diagonal weights improves the Weiss approximation for paramagnetic problems, but not for spin glass problems.

## 1 Introduction

Boltzmann Machines (BMs) [1], are networks of binary neurons with a stochastic neuron dynamics, known as Glauber dynamics. Assuming symmetric connections between neurons, the probability distribution over neuron states $\vec{s}$ will become stationary and is given by the Boltzmann-Gibbs distribution $P(\vec{s})$. The Boltzmann distribution is a known function of the weights and thresholds of the network. However, computation of $P(\vec{s})$ or any statistics involving $P(\vec{s})$, such as mean firing rates or correlations, requires exponential time in the number of neurons. This is

due to the fact that $P(\vec{s})$ contains a normalization term $Z$, which involves a sum over all states in the network, of which there are exponentially many. This problem is particularly important for BM learning.

Using statistical sampling techiques [2], learning can be significantly improved [1]. However, the method has rather poor convergence and can only be applied to small networks.

In [3, 4], an acceleration method for learning in BMs is proposed using mean field theory by replacing $\langle s_i s_j \rangle$ by $m_i m_j$ in the learning rule. It can be shown [5] that such a naive mean field approximation of the learning rules does not converge in general. Furthermore, we argue that the correlations can be computed using the linear response theorem [6].

In [7, 5] the mean field approximation is derived by making use of the properties of convex functions (Jensen's inequality and tangential bounds). In this paper we present an alternative derivation which uses a Legendre transformation and a small coupling expansion [8]. It has the advantage that higher order contributions (TAP and higher) can be computed in a systematic manner and that it may be applicable to arbitrary graphical models.

## 2 Boltzmann Machine learning

The Boltzmann Machine is defined as follows. The possible configurations of the network can be characterized by a vector $\vec{s} = (s_1, .., s_i, .., s_n)$, where $s_i = \pm 1$ is the state of the neuron $i$, and $n$ the total number of the neurons. Neurons are updated using Glauber dynamics.

Let us define the energy of a configuration $\vec{s}$ as

$$-E(\vec{s}) = \frac{1}{2} \sum_{i,j} w_{ij} s_i s_j + \sum_i s_i \theta_i.$$

After long times, the probability to find the network in a state $\vec{s}$ becomes independent of time (thermal equilibrium) and is given by the Boltzmann distribution

$$p(\vec{s}) = \frac{1}{Z} \exp\{-E(\vec{s})\}. \tag{1}$$

$Z = \sum_{\vec{s}} \exp\{-E(\vec{s})\}$ is the partition function which normalizes the probability distribution.

Learning [1] consists of adjusting the weights and thresholds in such a way that the Boltzmann distribution approximates a target distribution $q(\vec{s})$ as closely as possible.

A suitable measure of the difference between the distributions $p(\vec{s})$ and $q(\vec{s})$ is the Kullback divergence [9]

$$K = \sum_{\vec{s}} q(\vec{s}) \log \frac{q(\vec{s})}{p(\vec{s})}. \tag{2}$$

Learning consists of minimizing $K$ using gradient descent [1]

$$\Delta w_{ij} = \eta \left( \langle s_i s_j \rangle_c - \langle s_i s_j \rangle \right), \quad \Delta \theta_i = \eta \left( \langle s_i \rangle_c - \langle s_i \rangle \right).$$

The parameter $\eta$ is the learning rate. The brackets $\langle \cdot \rangle$ and $\langle \cdot \rangle_c$ denote the 'free' and 'clamped' expectation values, respectively.

The computation of both the free and the clamped expectation values is intractible, because it consists of a sum over all unclamped states. As a result, the BM learning algorithm can not be applied to practical problems.

## 3  The mean field approximation

We derive the mean field free energy using the small $\gamma$ expansion as introduced by Plefka [8]. The energy of the network is given by

$$E(s, w, h, \gamma) = \gamma E_{\text{int}} - \sum_i \theta_i s_i$$

$$E_{\text{int}} = -\frac{1}{2} \sum_{ij} w_{ij} s_i s_j$$

for $\gamma = 1$. The free energy is given by

$$F(w, \theta, \gamma) = -\log \text{Tr}_s e^{-E(s, w, \theta, \gamma)}$$

and is a function of the independent variables $w_{ij}$, $\theta_i$ and $\gamma$. We perform a Legendre transformation on the variables $\theta_i$ by introducing $m_i = -\frac{\partial F}{\partial \theta_i}$. The Gibbs free energy

$$G(w, m, \gamma) = F(w, \theta, \gamma) + \sum_i \theta_i m_i$$

is now a function of the independent variables $m_i$ and $w_{ij}$, and $\theta_i$ is implicitly given by $\langle s_i \rangle_\gamma = m_i$. The expectation $\langle \cdot \rangle_\gamma$ is with respect to the full model with interaction $\gamma$.

We expand

$$G(\gamma) = G(0) + \gamma G'(0) + \frac{1}{2}\gamma^2 G''(0) + \mathcal{O}(\gamma^3)$$

We directly obtain from [8]

$$G'(\gamma) = \langle E_{\text{int}} \rangle_\gamma$$

$$G''(\gamma) = \langle E_{\text{int}} \rangle_\gamma^2 - \langle E_{\text{int}}^2 \rangle_\gamma + \left\langle E_{\text{int}} \sum_i \frac{\partial \theta_i}{\partial \gamma}(s_i - m_i) \right\rangle_\gamma$$

For $\gamma = 0$ the expectation values $\langle \cdot \rangle_\gamma$ become the mean field expectations which we can directly compute:

$$G(0) = \frac{1}{2} \sum_i \left( (1 + m_i) \log \frac{1}{2}(1 + m_i) + (1 - m_i) \log \frac{1}{2}(1 - m_i) \right)$$

$$G'(0) = -\frac{1}{2} \sum_{ij} w_{ij} m_i m_j$$

$$G''(0) = -\frac{1}{4} \sum_{ij} w_{ij}^2 (1 - m_i^2)(1 - m_j^2)$$

Thus

$$G(1) = \frac{1}{2} \sum_i \left( (1 + m_i) \log \frac{1}{2}(1 + m_i) + (1 - m_i) \log \frac{1}{2}(1 - m_i) \right)$$
$$\quad - \frac{1}{2} \sum_{ij} w_{ij} m_i m_j$$
$$\quad - \frac{1}{2} \sum_{ij} w_{ij}^2 (1 - m_i^2)(1 - m_j^2) + \mathcal{O}(w^3 f(m)) \qquad (3)$$

where $f(m)$ is some unknown function of $m$.

The mean field equations are given by the inverse Legendre transformation

$$\theta_i = \frac{\partial G}{\partial m_i} = \tanh^{-1}(m_i) - \sum_j w_{ij} m_j + \sum_j w_{ij}^2 m_i (1 - m_j^2), \qquad (4)$$

which we recognize as the mean field equations.

The correlations are given by

$$\langle s_i s_j \rangle - \langle s_i \rangle \langle s_j \rangle = -\frac{\partial^2 F}{\partial \theta_i \partial \theta_j} = \frac{\partial m_i}{\partial \theta_j} = \left( \frac{\partial \theta}{\partial m} \right)_{ij}^{-1} = \left( \frac{\partial^2 G}{\partial m^2} \right)_{ij}^{-1}.$$

We therefore obtain from Eq. 3

$$\langle s_i s_j \rangle - \langle s_i \rangle \langle s_j \rangle = A_{ij}$$

with

$$(A^{-1})_{ij} = \delta_{ij} \left( \frac{1}{1 - m_i^2} + \sum_k w_{ik}^2 (1 - m_k^2) \right) - w_{ij} - 2 m_i m_j w_{ij}^2 \qquad (5)$$

Thus, for given $w_{ij}$ and $\theta_i$, we obtain the approximate mean firing rates $m_i$ by solving Eqs. 4 and the correlations by their linear response approximations Eqs. 5. The inclusion of hidden units is straigthforward. One applies the above approximations in the free and the clamped phase separately [5]. The complexity of the method is $O(n^3)$, due to the matrix inversion.

## 4 Learning without hidden units

We will assess the accuracy of the above method for networks without hidden units. Let us define $C_{ij} = \langle s_i s_j \rangle_c - \langle s_i \rangle_c \langle s_j \rangle_c$, which can be directly computed from the data. The fixed point equation for $\Delta \theta_i$ gives

$$\Delta \theta_i = 0 \Leftrightarrow m_i = \langle s_i \rangle_c. \qquad (6)$$

The fixed point equation for $\Delta w_{ij}$, using Eq. 6, gives

$$\Delta w_{ij} = 0 \Leftrightarrow A_{ij} = C_{ij}, i \neq j. \qquad (7)$$

From Eq. 7 and Eq. 5 we can solve for $w_{ij}$, using a standard least squares method. In our case, we used **fsolve** from Matlab. Subsequently, we obtain $\theta_i$ from Eq. 4. We refer to this method as the TAP approximation.

In order to assess the effect of the TAP term, we also computed the weights and thresholds in the same way as described above, but without the terms of order $w^2$ in Eqs. 5 and 4. Since this is the standard Weiss mean field expression, we refer to this method as the Weiss approximation.

The fixed point equations are only imposed for the off-diagonal elements of $\Delta w_{ij}$ because the Boltzmann distribution Eq. 1 does not depend on the diagonal elements $w_{ii}$. In [5], we explored a variant of the Weiss approximation, where we included diagonal weight terms. As is discussed there, if we were to impose Eq. 7 for $i = j$ as well, we have $A = C$. If $C$ is invertible, we therefore have $A^{-1} = C^{-1}$. However, we now have more constraints than variables. Therefore, we introduce diagonal weights $w_{ii}$ by adding the term $w_{ii} m_i$ to the righthandside of Eq. 4 in the Weiss approximation. Thus,

$$w_{ij} = \frac{\delta_{ij}}{1 - m_i^2} - (C^{-1})_{ij}$$

and $\theta_i$ is given by Eq. 4 in the Weiss approximation. Clearly, this method is computationally simpler because it gives an explicit expression for the solution of the weights involving only one matrix inversion.

## 5   Numerical results

For the target distribution $q(s)$ in Eq. 2 we chose a fully connected Ising spin glass model with equilibrium distribution

$$q(s) = \frac{1}{Z} \exp\{-\frac{1}{2} \sum_{ij} J_{ij} s_i s_j\}$$

with $J_{ij}$ i.i.d. Gaussian variables with mean $\frac{J_0}{n-1}$ and variance $\frac{J^2}{n-1}$. This model is known as the Sherrington-Kirkpatrick (SK) model [10]. Depending on the values of $J$ and $J_0$, the model displays a para-magnetic (unordered), ferro-magnetic (ordered) and a spin-glass (frustrated) phase. For $J_0 = 0$, the para-magnetic (spin-glass) phase is obtained for $J < 1$ ($J > 1$). We will assess the effectiveness of our approximations for finite $n$, for $J_0 = 0$ and for various values of $J$. Since this is a realizable task, the optimal KL divergence is zero, which is indeed observed in our simulations.

We measure the quality of the solutions by means of the Kullback divergence. Therefore, this comparison is only feasible for small networks. The reason is that the computation of the Kullback divergence requires the computation of the Boltzmann distribution, Eq. 1, which requires exponential time due to the partition function $Z$.

We present results for a network of $n = 10$ neurons. For $J_0 = 0$, we generated for each value of $0.1 < J < 3$, 10 random weight matrices $J_{ij}$. For each weight matrix, we computed the $q(\vec{s})$ on all $2^n$ states. For each of the 10 problems, we applied the TAP method, the Weiss method and the Weiss method with diagonal weights. In addition, we applied the exact Boltzmann Machine learning algorithm using conjugate gradient descent and verified that it gives KL divergence equal to zero, as it should. We also applied a factorized model $p(\vec{s}) = \prod_i \frac{1}{2}(1 + m_i s_i)$ with $m_i = \langle s_i \rangle_c$ to assess the importance of correlations in the target distribution. In Fig. 1a, we show for each $J$ the average KL divergence over the 10 problem instances as a function of $J$ for the TAP method, the Weiss method, the Weiss method with diagonal weights and the factorized model. We observe that the TAP method gives the best results, but that its performance deteriorates in the spin-glass phase ($J > 1$).

The behaviour of all approximate methods is highly dependent on the individual problem instance. In Fig. 1b, we show the mean value of the KL divergence of the TAP solution, together with the minimum and maximum values obtained on the 10 problem instances.

Despite these large fluctuations, the quality of the TAP solution is consistently better than the Weiss solution. In Fig. 1c, we plot the difference between the TAP and Weiss solution, averaged over the 10 problem instances.

In [5] we concluded that the Weiss solution with diagonal weights is better than the standard Weiss solution when learning a finite number of randomly generated patterns. In Fig. 1d we plot the difference between the Weiss solution with and without diagonal weights. We observe again that the inclusion of diagonal weights leads to better results in the paramagnetic phase ($J < 1$), but leads to worse results in the spin-glass phase. For $J > 2$, we encountered problem instances for which either the matrix $C$ is not invertible or the KL divergence is infinite. This problem becomes more and more severe for increasing $J$. We therefore have not presented results for the Weiss approximation with diagonal weigths for $J > 2$.

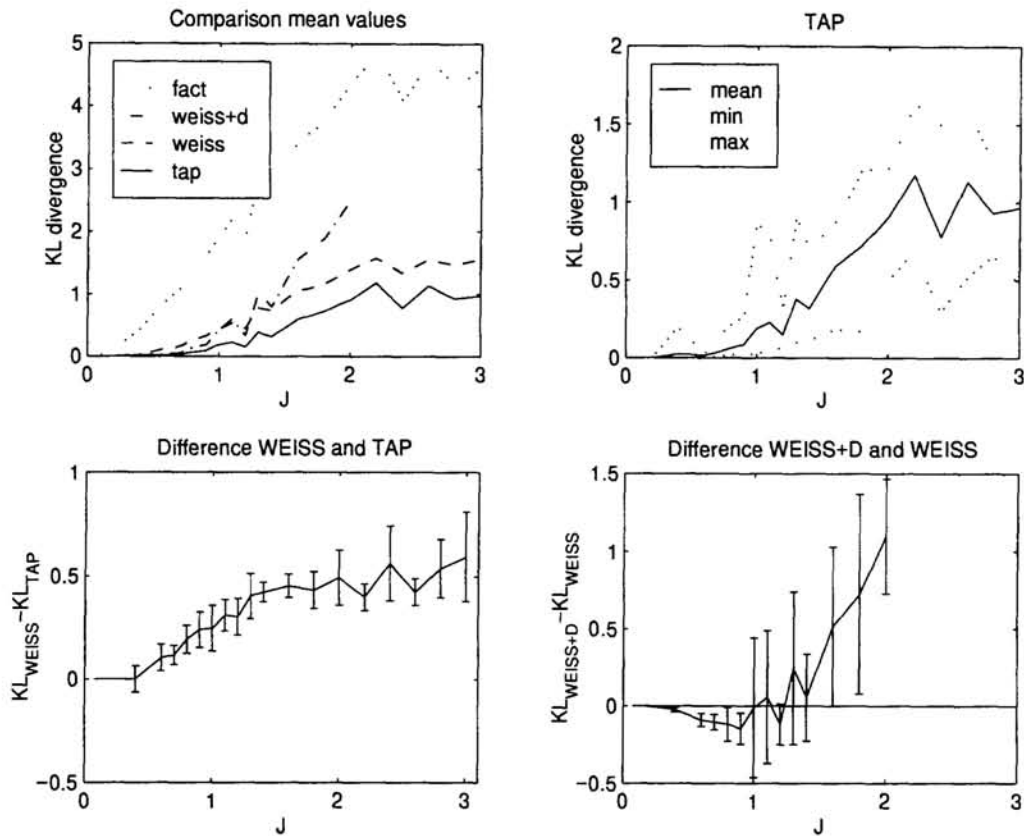

Figure 1: Mean field learning of paramagnetic ($J < 1$) and spin glass ($J > 1$) problems for a network of 10 neurons. a) Comparison of mean KL divergences for the factorized model (fact), the Weiss mean field approximation with and without diagonal weights (weiss+d and weiss), and the TAP approximation, as a function of $J$. The exact method yields zero KL divergence for all $J$. b) The mean, minimum and maximum KL divergence of the TAP approximation for the 10 problem instances, as a function of $J$. c) The mean difference between the KL divergence for the Weiss approximation and the TAP approximation, as a function of $J$. d) The mean difference between the KL divergence for the Weiss approximation with and without diagonal weights, as a function of $J$.

## 6   Discussion

We have presented a derivation of mean field theory and the linear response correction based on a small coupling expansion of the Gibbs free energy. This expansion can in principle be computed to arbitrary order. However, one should expect that the solution of the resulting mean field and linear response equations will become more and more difficult to solve numerically. The small coupling expansion should be applicable to other network models such as the sigmoid belief network, Potts networks and higher order Boltzmann Machines.

The numerical results show that the method is applicable to paramagnetic problems. This is intuitively clear, since paramagnetic problems have a unimodal probability distribution, which can be approximated by a mean and correlations around the mean. The method performs worse for spin glass problems. However, it still gives a useful approximation of the correlations when compared to the factorized model which ignores all correlations. In this regime, the TAP approximation improves

significantly on the Weiss approximation. One may therefore hope that higher order approximation may further improve the method for spin glass problems. Therefore, we cannot conclude at this point whether mean field methods are restricted to unimodal distributions. In order to further investigate this issue, one should also study the ferromagnetic case ($J_0 > 1, J > 1$), which is multimodal as well but less challenging than the spin glass case.

It is interesting to note that the performance of the exact method is absolutely insensitive to the value of $J$. Naively, one might have thought that for highly multi-modal target distributions, any gradient based learning method will suffer from local minima. Apparently, this is not the case: the exact KL divergence has just one minimum, but the mean field approximations of the gradients may have multiple solutions.

## Acknowledgement

This research is supported by the Technology Foundation STW, applied science division of NWO and the techology programme of the Ministry of Economic Affairs.

## References

[1] D. Ackley, G. Hinton, and T. Sejnowski. A learning algorithm for Boltzmann Machines. *Cognitive Science*, 9:147–169, 1985.

[2] C. Itzykson and J-M. Drouffe. *Statistical Field Theory*. Cambridge monographs on mathematical physics. Cambridge University Press, Cambridge, UK, 1989.

[3] C. Peterson and J.R. Anderson. A mean field theory learning algorithm for neural networks. *Complex Systems*, 1:995–1019, 1987.

[4] G.E. Hinton. Deterministic Boltzmann learning performs steepest descent in weight-space. *Neural Computation*, 1:143–150, 1989.

[5] H.J. Kappen and F.B. Rodríguez. Efficient learning in Boltzmann Machines using linear response theory. *Neural Computation*, 1997. In press.

[6] G. Parisi. *Statistical Field Theory*. Frontiers in Physics. Addison-Wesley, 1988.

[7] L.K. Saul, T. Jaakkola, and M.I. Jordan. Mean field theory for sigmoid belief networks. *Journal of artificial intelligence research*, 4:61–76, 1996.

[8] T. Plefka. Convergence condition of the TAP equation for the infinite-range Ising spin glass model. *Journal of Physics A*, 15:1971–1978, 1982.

[9] S. Kullback. *Information Theory and Statistics*. Wiley, New York, 1959.

[10] D. Sherrington and S. Kirkpatrick. Solvable model of Spin-Glass. *Physical review letters*, 35:1792–1796, 1975.